# Resolving Perceptual Aliasing In The Presence Of Noisy Sensors*

**Ronen I. Brafman & Guy Shani**
Department of Computer Science
Ben-Gurion University
Beer-Sheva 84105, Israel
{brafman, shanigu}@cs.bgu.ac.il

## Abstract

Agents learning to act in a partially observable domain may need to overcome the problem of perceptual aliasing – i.e., different states that appear similar but require different responses. This problem is exacerbated when the agent's sensors are noisy, i.e., sensors may produce different observations in the same state. We show that many well-known reinforcement learning methods designed to deal with perceptual aliasing, such as Utile Suffix Memory, finite size history windows, eligibility traces, and memory bits, do not handle noisy sensors well. We suggest a new algorithm, Noisy Utile Suffix Memory (NUSM), based on USM, that uses a weighted classification of observed trajectories. We compare NUSM to the above methods and show it to be more robust to noise.

## 1 Introduction

Consider an agent situated in a partially observable domain: It executes an action; the action may change the state of the world; this change is reflected, in turn, by the agent's sensors; the action may have some associated cost, and the new state may have some associated reward or penalty. Thus, the agent's interaction with this environment is characterized by a sequence of action-observation-reward steps, known as *instances* [7]. In this paper we are interested in agents with imperfect and noisy sensors that learn to act in such environments without any prior information about the underlying set of world-states and the world's dynamics, only information about their sensor's capabilities. This is a known variant of reinforcement learning (RL) in partially observable domains [1].

As the agent works with observations, rather than states, two possible problems arise: The agent may observe too much data which requires computationally intensive filtering – a problem we do not discuss. Or, sensors may supply insufficient data to identify the current state of the world based on the current observation. This leads to a phenomena known as *perceptual aliasing* [2], where the same observation is obtained in distinct states requiring different actions. For example, in Figure 1(a) the states marked with $X$ are perceptually aliased. Various RL techniques were developed to handle this problem.

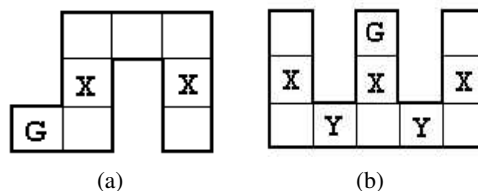

Figure 1: Two maze domains. If only wall configuration is sensed, states marked with the same letter (X or Y) are perceptually aliased.

The problem of resolving the perceptual aliasing is exacerbated when the agent's sensors are not deterministic. For example, if walls can sometimes be reported where none exist, or if a wall sometimes goes undetected. The performance of existing techniques for handling RL domains with perceptually aliased states, such as finite size history windows, eligibility traces, and internal memory quickly degrades as the level of noise increases.

In this paper, we introduce the Noisy Utile Suffix Memory (MUSM) algorithm, which builds on McCallum's Utile Suffix Memory (USM) algorithm [7]. We compare the performance of MUSM to USM and other existing methods on the above two standard maze domains, and show that it is more robust to noise.

## 2   Background

We briefly review a number of algorithms for resolving the problem of perceptual aliasing. We assume familiarity with basic RL techniques, and in particular, Q-learning, SARSA, and eligibility traces (see Sutton and Barto [12] for more details).

The simplest way to handle perceptual aliasing is to ignore it by using the observation space as the state space, defining a *memory-less* policy. This approach works sometimes, although it generally results in poor performance. Jaakkola *et al.* [4] suggest stochastic memory-less policies, and Williams and Singh [13] implement an online version of their algorithm and showed that it approximates an optimal solution with $50\%$ accuracy.

Eligibility traces can be viewed as a type of short-term memory, as they update the recently visited state-action couplings. Therefore, they can be used to augment the ability to handle partial observability. Loch and Singh [6] explore problems where a memory-less optimal policy exists and demonstrated that Sarsa($\lambda$) can learn an optimal policy for such domains.

Finite-size history methods are a natural extension to memory-less policies. Instead of identifying a state with the last observation, we can use the last $k$ observations. The size of the history window ($k$) is a fixed parameter, identical for all observation sequences. Lock and Singh [6] show Sarsa($\lambda$) using fixed history window to learn a good policy in domains where short-term memory optimal policies exist.

An arbitrarily predefined history window cannot generally solve perceptual aliasing: an agent cannot be expected to know in advance how long should it remember the action-observation-reward trajectories. Usually, some areas of the state space require the agent to remember more, while in other locations a reactive policy is sufficient. A better solution is to learn online the history length needed to decide on the best action in the current location. McCallum [7] extensively handles those issues in his dissertation. We review McCallums' Utile Suffix Memory (USM) algorithm in Section 3.

Another possible approach to handle perceptual aliasing is to augment the observations with some internal memory [10]. The agents' state $s$ is composed of both the current

observation $o$ and the internal memory state $m$. The agents' actions are enhanced with actions that modify the memory state by flipping one of the bits. The agent uses some standard learning mechanism to learn the proper action, including actions that change the memory state. The algorithm is modified so that no cost or decay is applied to the actions that modify the memory. This approach is better than using finite or variable history length because meaningful events can occur arbitrarily far in the past. Keeping all the possible trajectories from the event onwards until the outcome is observed might cost too much, and McCallums' techniques are unable to group those trajectories together to deduce the proper action. Peshkin *et al.* [10] demonstrated the memory-bits approach to converge, but did not show it to be superior to any other algorithm.

Other approaches include the use of finite-state automata (FSA) [8] (which can be viewed as a special case of the memory-bits approach); the use of neural networks for internal memory [5, 3]; and constructing and solving a POMDP model [7, 2]. The new emerging technique of Predictive State Representations (PSR) [11] may also provide a good way to online learn a model of the environment. We note that most researchers tested their algorithms on environments with very little noise, and do not examine the effect of noise on their performance.

## 3 Utile Suffix Memory

Instance-based state identification [7] resolves perceptual aliasing with variable length short term memory. An Instance is a tuple $T_t = \langle T_{t-1}, a_{t-1}, o_t, r_t \rangle$ — the individual observed raw experience. Algorithms of this family keep all the observed raw data (sequences of instances), and use it to identify matching subsequences. The algorithm assumes that if the suffix of two sequences is similar both were likely generated in the same world state.

Utile Suffix Memory creates a tree structure, based on the well known suffix trees for string operations. This tree maintains the raw experiences and identifies matching suffixes. The root of the tree is an unlabelled node, holding all available instances. Each immediate child of the root is labelled with one of the observations encountered during the test. A node holds all the instances $T_t = \langle T_{t-1}, a_{t-1}, o_t, r_t \rangle$ whose final observation $o_t$ matches the observation in the node's label. At the next level, instances are split based on the last action of the instance $a_t$. Then, we split again based on (the next to last) observation $o_{t-1}$, etc. All nodes act as buckets, grouping together instances that have matching history suffixes of a certain length. Leaves take the role of states, holding $Q$-values and updating them. The deeper a leaf is in the tree, the more history the instances in this leaf share.

The tree is built on-line during the test run. To add a new instance to the tree, we examine its precept, and follow the path to the child node labeled by that precept. We then look at the action before this precept and move to the node labeled by that action, then branch on the precept prior to that action and so forth, until a leaf is reached.

Identifying the proper depth for a certain leaf is a major issue, and we shall present a number of improvements to McCallum's methods. Leaves should be split if their descendants show a statistical difference in expected future discounted reward associated with the same action. We split instances in a node if knowing where the agent came from helps predict future discounted rewards. Thus, the tree must keep what McCallum calls fringes, i.e., subtrees below the "official" leaves.

For better performance, McCallum did not compare the nodes in the fringes to their siblings, only to their ancestor "official" leaf. He also did not compare values from all actions executed from the fringe, only the action that has the highest $Q$-value in the leaf (the policy action of that leaf). To compare the populations of expected discounted future rewards from the two nodes (the fringe and the "official" leaf), he used the Kolmogorov-Smirnov (KS)

test — a non-parametric statistical test used to find whether two populations were generated by the same distribution. If the test reported that a difference was found between the expected discounted future rewards after executing the policy action, the leaf was split, the fringe node would become the new leaf, and the tree would be expanded to create deeper fringes. Figure 3 presents an example of a possible USM tree, without fringe nodes.

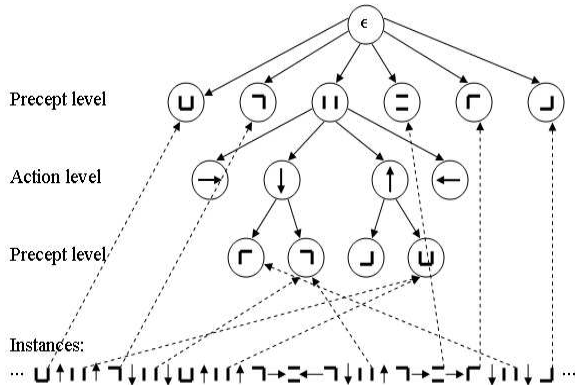

Figure 2: A possible USM suffix tree generated by the maze in Figure 1(a). Below is a sequence of instances demonstrating how some instances are clustered into the tree leaves.

Instead of comparing the fringe node to its ancestor "official" leaf, we found it computationally possible to compare the siblings of the fringe, avoiding the problem that the same instance appears in both distributions. McCallum compared only the expected discounted future rewards from executing the policy action, where we compare all the values following all actions executed after any of the instances in the fringe. McCallum used the KS test, where we choose to use the more robust randomization test [14] that works well with small sets of instances. McCallum also considered only fringe nodes of certain depth, given as a parameter to the algorithm, where we choose to create fringe nodes as deep as possible, until the number of instances in the node diminish below some threshold (we use a value of 10 in our experiments).

The expected future discounted reward of instance $T_i$ is defined by:

$$Q(T_i) = r_i + \gamma U(L(T_{i+1})) \tag{1}$$

where $L(T_i)$ is the leaf associated with instance $T_i$ and $U(s) = max_a(Q(s,a))$.

After inserting new instances to the tree, we update $Q$-values in the leaves using:

$$R(s,a) = \frac{\sum_{T_i \in T(s,a)} r_i}{|T(s,a)|} \tag{2}$$

$$Pr(s'|s,a) = \frac{|\forall T_i \in T(s,a), L(T_{i+1}) = s'|}{|T(s,a)|} \tag{3}$$

$$Q(s,a) = R(s,a) + \gamma \sum_{s'} Pr(s'|s,a)U(s') \tag{4}$$

We use $s$ and $s'$ to denote the leaves of the tree, as in an optimal tree configuration for a problem the leaves of the tree define the sates of the underlying MDP. The above equations therefore correspond to a single step of the value iteration algorithm used in MDPs.

Now that the $Q$-values have been updated, the agent chooses the next action to perform based on the $Q$-values in the leaf corresponding to the current instance $T_t$:

$$a_{t+1} = argmax_a Q(L(T_t), a) \tag{5}$$

McCallum uses the fringes of the tree for a smart exploration strategy. In our implementation we use a simple $\epsilon$-greedy technique for exploration.

## 4   Adding Noise to Utile Suffix Memory

There are two types of noise in perception. The system makes different observations at the same location (false negative), or it makes identical observations at different locations (false positive). USM handles false positives by differentiating identical current observations using the agents' history. Knowing that the agent came from different locations, helps it realize that it is in two different locations, though the observations look the same.

USM does not handle false negative perceptions well. When the agent is at the same state, but receives different observations, it is unable to learn from the noisy observation and thus wastes much of the available information. Our Noisy Utile Suffix Memory (NUSM) is designed to overcome this weakness.

It is reasonable to assume that the agent has some sensor model defining $pr(o|s)$ — the probability that the agent will observe $o$ in world state $s$. We can use the sensor model to augment USM with the ability to learn from noisy instances. In our experiments we assume the agent has $n$ boolean sensors with an accuracy probability $p_i$. A single observation is composed of $n$ output values $o = \langle \omega_1, ..., \omega_n \rangle$ when $\omega_i \in 0, 1$. The probability that an observation $o = \langle \omega_1^o, ..., \omega_1^o \rangle$ came from an actual world state $s = \langle \omega_1^s, ..., \omega_n^s \rangle$ is therefore:

$$pr(o|s) = \prod_{i=0}^{n} \Delta_i \tag{6}$$

$$\Delta_i = \begin{cases} p_i & \omega_i^0 = \omega_i^s \\ (1 - p_i) & \omega_i^0 \neq \omega_i^s \end{cases} \tag{7}$$

USM inserts each instance into a single path, ending at one leaf. Using any sensor model we can insert the new instance $T_t = \langle T_{t-1}, a_{t-1}, o_t, r_t \rangle$ into several paths with different weights. When inserting $T_t$ with weight $w$ into an action node at depth $k$ (with its children labeled by observations) we will insert the instance into every child node $c$, with weight $w \cdot pr(o_{t-k-1}|label(c))$. When inserting $T_t$ with weight $w$ into an observation node at depth $k$ (with its children labeled by actions) we will insert the instance only into the child $c$ labeled by $a_{t-k-1}$ with the same weight $w$. Weights of instances are stored in each node with the instance as $w_s(T_t)$ — the weight of instance $T_t$ in node $s$.

We can now rewrite Equation 2 and Equation 3:

$$R(s, a) = \frac{\sum_{T_i \in T(s,a)} r_i \cdot w_s(T_i)}{\sum_{T_i \in T(s,a)} w_s(T_i)} \tag{8}$$

$$Pr(s'|s, a) = \frac{\sum_{T_i, L(T_{i+1})=s'} w_s(T_i)}{\sum_{T_i \in T(s,a)} w_s(T_i)} \tag{9}$$

The noisy instances are used only for updating the $Q$-values in the leaves. The test for splitting is still calculated using identical sequences only. The tree structure of USM and NUSM is hence identical, but NUSM learns a better policy. Conducted experiments indicate that using the noisy sequences for deciding when to split leaves provides a slight gain in collected rewards, but the constructed tree is much larger, resulting in a considerable hit to performance.

NUSM learns noisy sequences better. When a state corresponding to a noisy sequence is observed, even though the noise in it might make it rare, NUSM can still use data from real sequences to decide which action is useful for this state.

# 5   Experimental Results

To test our algorithms we used two maze worlds, seen in Figure 1(a) and Figure 1(b), identical to the worlds McCallum used to show the performance of the USM algorithm. In both cases some of the world states are perceptually aliased and the algorithm should learn to identify the real world states. The agent in our experiments has four sensors allowing it to sense an immediate wall above, below, to the left, and to the right of its current location. Sensors have a boolean output that has a probability $\alpha$ of being accurate. The probability of all sensors providing the correct output is $\alpha^4$. In both mazes there is a single location that grants the agent a reward of 10. Upon receiving that reward the agent is transformed to any of the perceptually aliased states of the maze randomly. If the agent bumps into a wall it pays a cost (a negative reward) of 1. For every move the agent pays a cost of 0.1.

We compared the performance of applying finite size history windows to $Q$-learning and Sarsa, eligibility traces, memory bits, USM and NUSM on the two worlds. In the tables below, $QL_2$ denotes using $Q$-learning with a history window of size 2, and $S_2$ denotes using Sarsa with a window size of 2. $S(\lambda)$ denotes the Sarsa($\lambda$) algorithm. Adding the superfix 1 denotes adding 1 bit of internal memory. For example, $S(\lambda)_2^1$ denotes using Sarsa($\lambda$) with a history window of 2 and 1 internal memory bit. The columns and rows marked $\sigma^2$ present the average variance over methods (bottom row) and $\alpha$ values (rightmost column). In the NUSM column, in brackets, is the improvement NUSM gains over USM.

As we are only interested in the effect of noisy sensors, the maze examples we use do not demonstrate the advantages of the various algorithms; USM's ability to automatically compute the needed trajectory length in different locations and the internal memory ability to remember events that occurred arbitrary far in the past are unneeded since our examples require the agent to look only at the last 2 instances in every perceptually aliased location.

We ran each algorithm for 50000 steps learning a policy as explained above, and calculated the average reward over the last 5000 iterations only to avoid the difference in convergence time. We ran experiments with varying values of $\alpha$ (accuracy of the sensors) ranging from 1.00 (sensor output is without noise) to 0.85 (overall output accuracy of 0.522). Reported results are averaged over 10 different executions of each algorithm. We also ran experiments for Sarsa and $Q$-learning with only the immediate observation, which yielded poor results as expected, and for history window of size 3 and 4 which resulted in lower performance than history window of size 2 for all algorithms (and were therefore removed from the tables). Additional memory bits did not improve performance either. In our Sarsa($\lambda$) implementation we used $\lambda = 0.9$.

| $\alpha$ | $QL_2$ | $S_2$ | $S(\lambda)_1$ | $S(\lambda)_2$ | $S(\lambda)_3$ | $S(\lambda)_1^1$ | $S(\lambda)_2^1$ | USM | NUSM | $\sigma^2$ |
|---|---|---|---|---|---|---|---|---|---|---|
| 1.00 | 1.51 | 1.54 | 0.32 | 1.53 | 0.98 | 1.27 | 1.53 | 1.56 | **1.57**(+0%) | 0.033 |
| 0.99 | 1.46 | 1.47 | 0.33 | 1.47 | 0.98 | 1.34 | 1.45 | 1.49 | **1.54**(+3%) | 0.016 |
| 0.98 | 1.42 | 1.42 | 0.32 | 1.36 | 0.83 | 1.41 | 1.40 | 1.42 | **1.44**(+1%) | 0.008 |
| 0.97 | 1.36 | 1.38 | 0.41 | 1.31 | 0.64 | 1.35 | 1.35 | 1.38 | **1.43**(+4%) | 0.007 |
| 0.96 | 1.28 | 1.26 | 0.38 | 1.29 | 0.58 | 1.24 | 1.27 | 1.35 | **1.40**(+4%) | 0.014 |
| 0.95 | 1.24 | 1.23 | 0.43 | 1.21 | 0.55 | 1.26 | 1.22 | 1.30 | **1.35**(+4%) | 0.009 |
| 0.94 | 1.11 | 1.10 | 0.47 | 1.16 | 0.45 | 0.89 | 1.12 | 1.18 | **1.29**(+9%) | 0.025 |
| 0.93 | 1.05 | 1.03 | 0.42 | 1.13 | 0.43 | 0.94 | 1.07 | 1.16 | **1.29**(+11%) | 0.023 |
| 0.92 | 0.96 | 0.88 | 0.47 | 1.10 | 0.33 | 0.92 | 1.04 | 1.12 | **1.20**(+7%) | 0.014 |
| 0.91 | 0.88 | 0.82 | 0.47 | 1.06 | 0.29 | 0.74 | 0.94 | 0.96 | **1.12**(+17%) | 0.022 |
| 0.90 | 0.83 | 0.73 | 0.46 | 1.02 | 0.22 | 0.77 | 0.92 | 0.99 | **1.07**(+8%) | 0.013 |
| 0.89 | 0.74 | 0.60 | 0.48 | 0.95 | 0.23 | 0.80 | 0.87 | 0.93 | **1.04**(+12%) | 0.015 |
| 0.88 | 0.61 | 0.59 | 0.42 | 0.90 | 0.14 | 0.66 | 0.83 | 0.84 | **1.01**(+20%) | 0.013 |
| 0.87 | 0.58 | 0.50 | 0.48 | 0.85 | 0.12 | 0.63 | 0.78 | 0.71 | **0.98**(+38%) | 0.011 |
| 0.86 | 0.40 | 0.37 | 0.46 | 0.79 | 0.07 | 0.55 | 0.76 | 0.57 | **0.92**(+61%) | 0.021 |
| 0.85 | 0.45 | 0.35 | 0.45 | 0.75 | 0.05 | 0.47 | 0.68 | 0.47 | **0.87**(+85%) | 0.018 |
| $\sigma^2$ | 0.01 | 0.015 | 0.004 | 0.004 | 0.022 | 0.061 | 0.005 | 0.02 | 0.003 | |

Table 1: Average reward as function of sensor accuracy, for the maze in Figure 1(a).

| $\alpha$ | $QL_2$ | $S_2$ | $S(\lambda)_1$ | $S(\lambda)_2$ | $S(\lambda)_3$ | $S(\lambda)_1^1$ | $S(\lambda)_2^1$ | USM | NUSM | $\sigma^2$ |
|---|---|---|---|---|---|---|---|---|---|---|
| 1.00 | 1.42 | 1.46 | 0.23 | 1.53 | 1.49 | 1.47 | 1.54 | 1.75 | **1.72**(-2%) | 0.004 |
| 0.99 | 1.40 | 1.41 | 0.24 | 1.44 | 1.34 | 1.24 | 1.43 | 1.57 | **1.61**(+3%) | 0.027 |
| 0.98 | 1.33 | 1.35 | 0.25 | 1.35 | 1.24 | 0.94 | 1.34 | 1.43 | **1.46**(+2%) | 0.034 |
| 0.97 | 1.30 | 1.29 | 0.26 | 1.22 | 1.15 | 0.90 | 1.21 | 1.40 | **1.40**(0%) | 0.032 |
| 0.96 | 1.26 | 1.25 | 0.24 | 1.06 | 1.06 | 0.43 | 1.12 | 1.28 | **1.31**(+2%) | 0.015 |
| 0.95 | 1.19 | 1.16 | 0.21 | 1.00 | 0.90 | 0.33 | 1.03 | 1.23 | **1.26**(+2%) | 0.015 |
| 0.94 | 1.09 | 1.05 | 0.14 | 0.93 | 0.85 | 0.30 | 0.93 | 1.09 | **1.14**(+5%) | 0.011 |
| 0.93 | 1.05 | 0.94 | 0.12 | 0.82 | 0.76 | 0.39 | 0.77 | 1.09 | **1.09**(+0%) | 0.018 |
| 0.92 | 0.94 | 0.84 | 0.12 | 0.72 | 0.64 | 0.30 | 0.66 | 1.02 | **1.03**(+1%) | 0.011 |
| 0.91 | 0.85 | 0.80 | 0.09 | 0.77 | 0.47 | 0.24 | 0.48 | 0.93 | **0.96**(+3%) | 0.013 |
| 0.90 | 0.69 | 0.72 | 0.10 | 0.65 | 0.42 | 0.23 | 0.48 | 0.87 | **0.91**(+5%) | 0.013 |
| 0.89 | 0.64 | 0.58 | 0.06 | 0.58 | 0.24 | 0.24 | 0.40 | 0.81 | **0.92**(+14%) | 0.010 |
| 0.87 | 0.44 | 0.42 | 0.06 | 0.47 | 0.16 | 0.16 | 0.31 | 0.72 | **0.82**(+14%) | 0.012 |
| 0.86 | 0.30 | 0.21 | 0.04 | 0.26 | 0.09 | 0.13 | 0.20 | 0.68 | **0.81**(+19%) | 0.015 |
| 0.85 | 0.08 | 0.10 | 0.01 | 0.29 | 0.09 | 0.18 | 0.16 | 0.61 | **0.75**(+23%) | 0.008 |
| $\sigma^2$ | 0.008 | 0.01 | 0.003 | 0.009 | 0.006 | 0.071 | 0.016 | 0.011 | 0.006 | |

Table 2: Average reward as function of sensor accuracy, for the maze in Figure 1(b).

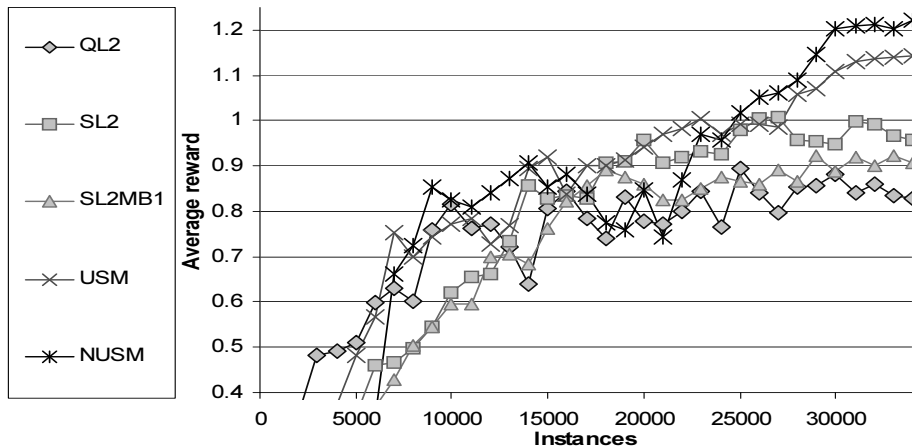

Figure 3: Convergence rates for the maze in Figure 1(a) when sensor accuracy is 0.9.

As we can see, when sensor output is only slightly noisy, all algorithms perform reasonably; NUSM performs the best, but differences are minor. This is because when sensors supply perfect output, resolving the perceptual aliasing results in a fully observable environment which $Q$-learning and Sarsa can solve optimally. When noise increases, the ability of NUSM to use similar suffixes of trajectories, results in a noticeable performance gain over other algorithms. The only algorithm that competes with NUSM is Sarsa($\lambda$) with a history window of 2. The ability of Sarsa($\lambda$) to perform well in partially observable domains have been noted by Lock and Singh [6][1], but we note here that the performance of Sarsa($\lambda$) relies heavily on the proper definition of the required history window size. When the history window differs slightly from the required size, the performance hit is substantial, as we can see in the two adjacent columns.

NUSM is more expensive computationally than USM and takes longer to converge. In Figure 3, we can see that it still converges reasonably fast. Moreover, each NUSM iteration takes about 5.6 milliseconds, when a USM iteration takes 3.1 milliseconds with the same

accuracy of $\alpha = 0.85$ and a similar number of nodes ($10, 229$ for NUSM and $10, 349$ for USM — including fringe nodes), making NUSM reasonable for online learning.

Finally, Both USM and NUSM attempt to disambiguate the perceptual aliasing and create a fully observable MDP. Yet, it is better to model the world directly as partially observable using a Partially Observable Markov Decision Process (POMDP). POMDP policies explicitly address the problem of incomplete knowledge, taking into account the ability of certain actions to reduce uncertainty without immediately generating useful rewards. Nikovski [9] used McCallum's Nearest Sequence Memory (NSM), a predecessor of USM to generate and solve a POMDP from the observed instances. They, however, considered environments with little noise. We implemented their algorithms and obtained poor results in the presence of noise in our domains, probably due to the use of NSM for state identification.

## 6 Conclusions

We show that some RL algorithms, including finite size history windows, memory bits and USM, that resolve perceptual aliasing, provide poor results in the presence of noisy sensors. We provided some insights as to why McCallums' USM algorithm does not handle well noisy input from the agents' sensors and introduce NUSM — an extension to USM that learns from noisy sequences and handles environments where sensors provide noisy output better. As noise arises, NUSM works better than other algorithms used for handling domains with perceptual aliasing.

## Footnotes

*Partially supported by the Lynn and William Frankel Center for Computer Sciences.

[1]Lock and Singh also recommend the use of replacing traces but we found that using accumulating traces produced better performance.

## References

[1] A. R. Cassandra, L. P. Kaelbling, and M. L. Littman. Acting optimally in partially observable stochastic domains. In *AAAI'94*, pages 1023–1028, 1994.

[2] L. Chrisman. Reinforcement learning with perceptual aliasing: The perceptual distinctions approach. In *AAAI'02*, pages 183–188, 1992.

[3] S. Hochreiter and J. Schmidhuber. Long short-term memory. *Neural Computation*, 9(8):1735–1780, 1997.

[4] T. Jaakkola, S. P. Singh, and M. I. Jordan. Reinforcement learning algorithm for partially observable Markov decision problems. In *NIPS'95*, volume 7, pages 345–352, 1995.

[5] L.-J. Lin and T. M. Mitchell. Memory approaches to reinforcement learning in non-markovian domains. Technical Report CMU-CS-92-138, 1992.

[6] J. Loch and S. Singh. Using eligibility traces to find the best memoryless policy in partially observable Markov decision processes. In *ICML'98*, pages 323–331, 1998.

[7] A. K. McCallum. *Reinforcement Learning with Selective Perception and Hidden State*. PhD thesis, University of Rochester, 1996.

[8] N. Meuleau, L. Peshkin, K. Kim, and L. P. Kaelbling. Learning finite-state controllers for partially observable environments. In *UAI'99*, pages 427–436, 1999.

[9] D. Nikovski. *State-Aggregation Algorithms for Learning Probabilistic Models for Robot Control*. PhD thesis, Carnegie Mellon University, 2002.

[10] L. Peshkin, N. Meuleau, and L. P. Kaelbling. Learning policies with external memory. In *ICML'99*, pages 307–314, 1999.

[11] S. Singh, M. L. Littman, and R. S. Sutton. Predictive representations of state. In *NIPS 2001*, pages 1555–1561, December 2001.

[12] R. S. Sutton and A. G. Barto. *Reinforcement Learning: An Introduction*. MIT Press, 1998.

[13] J. K. Williams and S. Singh. Experimental results on learning stochastic memoryless policies for partially observable markov decision processes. In *NIPS*, 1998.

[14] A. Yeh. More accurate tests for the statistical significance of result differences. In *18th Int. Conf. on Computational Linguistics*, pages 947–953, 2000.
